# STORING COVARIANCE BY THE ASSOCIATIVE LONG-TERM POTENTIATION AND DEPRESSION OF SYNAPTIC STRENGTHS IN THE HIPPOCAMPUS

Patric K. Stanton[*] and Terrence J. Sejnowski[†]
Department of Biophysics
Johns Hopkins University
Baltimore, MD 21218

## ABSTRACT

In modeling studies of memory based on neural networks, both the selective enhancement and depression of synaptic strengths are required for efficient storage of information (Sejnowski, 1977a,b; Kohonen, 1984; Bienenstock et al, 1982; Sejnowski and Tesauro, 1989). We have tested this assumption in the hippocampus, a cortical structure of the brain that is involved in long-term memory. A brief, high-frequency activation of excitatory synapses in the hippocampus produces an increase in synaptic strength known as long-term potentiation, or LTP (Bliss and Lomo, 1973), that can last for many days. LTP is known to be Hebbian since it requires the simultaneous release of neurotransmitter from presynaptic terminals coupled with postsynaptic depolarization (Kelso et al, 1986; Malinow and Miller, 1986; Gustaffson et al, 1987). However, a mechanism for the persistent reduction of synaptic strength that could balance LTP has not yet been demonstrated. We studied the associative interactions between separate inputs onto the same dendritic trees of hippocampal pyramidal cells of field CA1, and found that a low-frequency input which, by itself, does not persistently change synaptic strength, can either increase (associative LTP) or decrease in strength (associative long-term depression or LTD) depending upon whether it is positively or negatively correlated in time with a second, high-frequency bursting input. LTP of synaptic strength is Hebbian, and LTD is anti-Hebbian since it is elicited by pairing presynaptic firing with postsynaptic hyperpolarization sufficient to block postsynaptic activity. Thus, associative LTP and associative LTD are capable of storing information contained in the covariance between separate, converging hippocampal inputs.

[*]Present address: Departments of Neuroscience and Neurology, Albert Einstein College of Medicine, 1410 Pelham Parkway South, Bronx, NY 10461 USA.

[†]Present address: Computational Neurobiology Laboratory, The Salk Institute, P.O. Box 85800, San Diego, CA 92138 USA.

# INTRODUCTION

Associative LTP can be produced in some hippocampal neurons when low-frequency, (Weak) and high-frequency (Strong) inputs to the same cells are simultaneously activated (Levy and Steward, 1979; Levy and Steward, 1983; Barrionuevo and Brown, 1983). When stimulated alone, a weak input does not have a long-lasting effect on synaptic strength; however, when paired with stimulation of a separate strong input sufficient to produce homosynaptic LTP of that pathway, the weak pathway is associatively potentiated. Neural network modeling studies have predicted that, in addition to this Hebbian form of plasticity, synaptic strength should be weakened when weak and strong inputs are anti-correlated (Sejnowski, 1977a,b; Kohonen, 1984; Bienenstock et al, 1982; Sejnowski and Tesauro, 1989). Evidence for heterosynaptic depression in the hippocampus has been found for inputs that are inactive (Levy and Steward, 1979; Lynch et al, 1977) or weakly active (Levy and Steward, 1983) during the stimulation of a strong input, but this depression did not depend on any pattern of weak input activity and was not typically as long-lasting as LTP.

Therefore, we searched for conditions under which stimulation of a hippocampal pathway, rather than its inactivity, could produce either long-term depression or potentiation of synaptic strengths, depending on the pattern of stimulation. The stimulus paradigm that we used, illustrated in Fig. 1, is based on the finding that bursts of stimuli at 5 Hz are optimal in eliciting LTP in the hippocampus (Larson and Lynch, 1986). A high-frequency burst (STRONG) stimulus was applied to Schaffer collateral axons and a low-frequency (WEAK) stimulus given to a separate subicular input coming from the opposite side of the recording site, but terminating on dendrites of the same population of CA1 pyramidal neurons. Due to the rhythmic nature of the strong input bursts, each weak input shock could be either superimposed on the middle of each burst of the strong input (IN PHASE), or placed symmetrically between bursts (OUT OF PHASE).

# RESULTS

Extracellular evoked field potentials were recorded from the apical dendritic and somatic layers of CA1 pyramidal cells. The weak stimulus train was first applied alone and did not itself induce long-lasting changes. The strong site was then stimulated alone, which elicited homosynaptic LTP of the strong pathway but did not significantly alter amplitude of responses to the weak input. When weak and strong inputs were activated IN PHASE, there was an associative LTP of the weak input synapses, as shown in Fig. 2a. Both the synaptic excitatory post-synaptic potential (e.p.s.p.) ($\Delta$e.p.s.p. = +49.8 ± 7.8%, n=20) and population action potential ($\Delta$spike = +65.4 ± 16.0%, n=14) were significantly enhanced for at least 60 min up to 180 min following stimulation.

In contrast, when weak and strong inputs were applied OUT OF PHASE, they elicited an associative long-term depression (LTD) of the weak input synapses, as shown in Fig. 2b. There was a marked reduction in the population spike (-46.5 ± 11.4%, n=10) with smaller decreases in the e.p.s.p. (-13.8 ± 3.5%, n=13). Note that the stimulus patterns applied to each input were identical in these two experiments, and only the relative

phase of the weak and strong stimuli was altered. With these stimulus patterns, synaptic strength could be repeatedly enhanced and depressed in a single slice, as illustrated in Fig 2c. As a control experiment to determine whether information concerning covariance between the inputs was actually a determinant of plasticity, we combined the in phase and out of phase conditions, giving both the weak input shocks superimposed on the bursts plus those between the bursts, for a net frequency of 10 Hz. This pattern, which resulted in zero covariance between weak and strong inputs, produced no net change in weak input synaptic strength measured by extracellular evoked potentials. Thus, the asso-

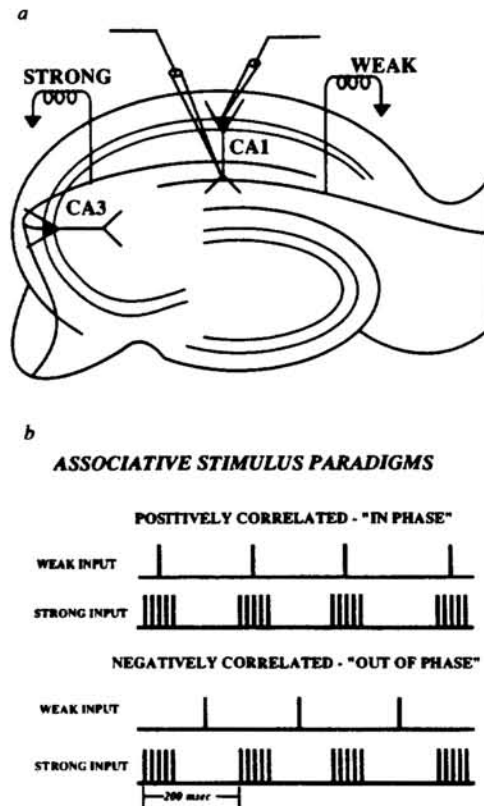

**Figure 1.** Hippocampal slice preparation and stimulus paradigms. **a:** The *in vitro* hippocampal slice showing recording sites in CA1 pyramidal cell somatic (stratum pyramidale) and dendritic (stratum radiatum) layers, and stimulus sites activating Schaffer collateral (**STRONG**) and commissural (**WEAK**) afferents. Hippocampal slices (400 μm thick) were incubated in an interface slice chamber at 34-35° C. Extracellular (1-5 MΩ resistance, 2M NaCl filled) and intracellular (70-120 MΩ, 2M K-acetate filled) recording electrodes, and bipolar glass-insulated platinum wire stimulating electrodes (50 μm tip diameter), were prepared by standard methods (Mody et al, 1988). **b:** Stimulus paradigms used. Strong input stimuli (**STRONG INPUT**) were four trains of 100 Hz bursts. Each burst had 5 stimuli and the interburst interval was 200 msec. Each train lasted 2 seconds for a total of 50 stimuli. Weak input stimuli (**WEAK INPUT**) were four trains of shocks at 5 Hz frequency, each train lasting for 2 seconds. When these inputs were **IN PHASE**, the weak single shocks were superimposed on the middle of each burst of the strong input. When the weak input was **OUT OF PHASE**, the single shocks were placed symmetrically between the bursts.

ciative LTP and LTD mechanisms appear to be balanced in a manner ideal for the storage of temporal covariance relations.

The simultaneous depolarization of the postsynaptic membrane and activation of glutamate receptors of the N-methyl-D-aspartate (NMDA) subtype appears to be necessary for LTP induction (Collingridge et al, 1983; Harris et al, 1984; Wigstrom and Gustaffson, 1984). The spread of current from strong to weak synapses in the dendritic tree,

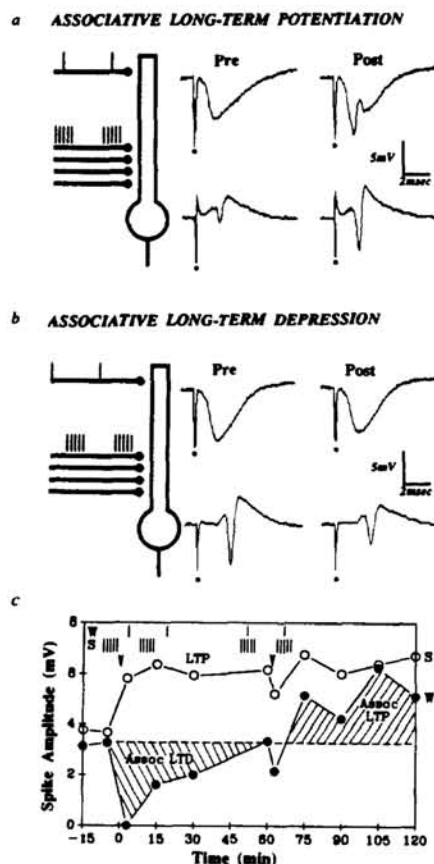

**Figure 2.** Illustration of associative long-term potentiation (LTP) and associative long-term depression (LTD) using extracellular recordings. **a:** Associative LTP of evoked excitatory postsynaptic potentials (e.p.s.p.'s) and population action potential responses in the weak input. Test responses are shown before (**Pre**) and 30 min after (**Post**) application of weak stimuli in phase with the coactive strong input. **b:** Associative LTD of evoked e.p.s.p.'s and population spike responses in the weak input. Test responses are shown before (**Pre**) and 30 min after (**Post**) application of weak stimuli out of phase with the coactive strong input. **c:** Time course of the changes in population spike amplitude observed at each input for a typical experiment. Test responses from the strong input (S, open circles), show that the high-frequency bursts (5 pulses/100 Hz, 200 msec interburst interval as in Fig. 1) elicited synapse-specific LTP independent of other input activity. Test responses from the weak input (W, filled circles) show that stimulation of the weak pathway out of phase with the strong one produced associative LTD (**Assoc LTD**) of this input. Associative LTP (**Assoc LTP**) of the same pathway was then elicited following in phase stimulation. Amplitude and duration of associative LTD or LTP could be increased by stimulating input pathways with more trains of shocks.

coupled with release of glutamate from the weak inputs, could account for the ability of the strong pathway to associatively potentiate a weak one (Kelso et al, 1986; Malinow and Miller, 1986; Gustaffson et al, 1987). Consistent with this hypothesis, we find that the NMDA receptor antagonist 2-amino-5-phosphonovaleric acid (AP5, 10 μM) blocks induction of associative LTP in CA1 pyramidal neurons (data not shown, n=5). In contrast, the application of AP5 to the bathing solution at this same concentration had no significant effect on associative LTD (data not shown, n=6). Thus, the induction of LTD seems to involve cellular mechanisms different from associative LTP.

The conditions necessary for LTD induction were explored in another series of experiments using intracellular recordings from CA1 pyramidal neurons made using standard techniques (Mody et al, 1988). Induction of associative LTP (Fig 3; WEAK S+W IN PHASE) produced an increase in amplitude of the single cell evoked e.p.s.p. and a lowered action potential threshold in the weak pathway, as reported previously (Barrionuevo and Brown, 1983). Conversely, the induction of associative LTD (Fig. 3; WEAK S+W OUT OF PHASE) was accompanied by a long-lasting reduction of e.p.s.p. amplitude and reduced ability to elicit action potential firing. As in control extracellular experiments, the weak input alone produced no long-lasting alterations in intracellular e.p.s.p.'s or firing properties, while the strong input alone yielded specific increases of the strong pathway e.p.s.p. without altering e.p.s.p.'s elicited by weak input stimulation.

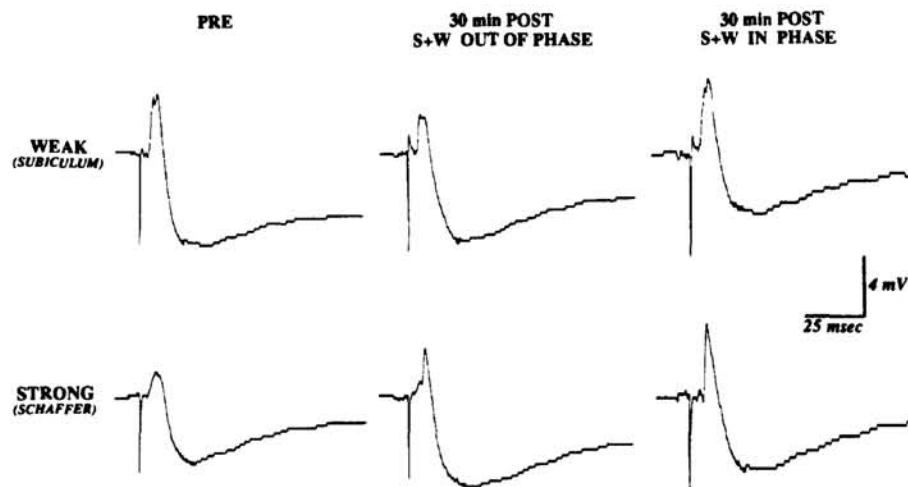

**Figure 3.** Demonstration of associative LTP and LTD using intracellular recordings from a CA1 pyramidal neuron. Intracellular e.p.s.p.'s prior to repetitive stimulation (**Pre**), 30 min after out of phase stimulation (**S+W OUT OF PHASE**), and 30 min after subsequent in phase stimuli (**S+W IN PHASE**). The strong input (Schaffer collateral side, lower traces) exhibited LTP of the evoked e.p.s.p. independent of weak input activity. Out of phase stimulation of the weak (Subicular side, upper traces) pathway produced a marked, persistent reduction in e.p.s.p. amplitude. In the same cell, subsequent in phase stimuli resulted in associative LTP of the weak input that reversed the LTD and enhanced amplitude of the e.p.s.p. past the original baseline. (RMP = -62 mV, $R_N$ = 30 MΩ)

A weak stimulus that is out of phase with a strong one arrives when the postsynaptic neuron is hyperpolarized as a consequence of inhibitory postsynaptic potentials and afterhyperpolarization from mechanisms intrinsic to pyramidal neurons. This suggests that postsynaptic hyperpolarization coupled with presynaptic activation may trigger LTD. To test this hypothesis, we injected current with intracellular microelectrodes to hyperpolarize or depolarize the cell while stimulating a synaptic input. Pairing the injection of depolarizing current with the weak input led to LTP of those synapses (Fig. 4a; STIM;

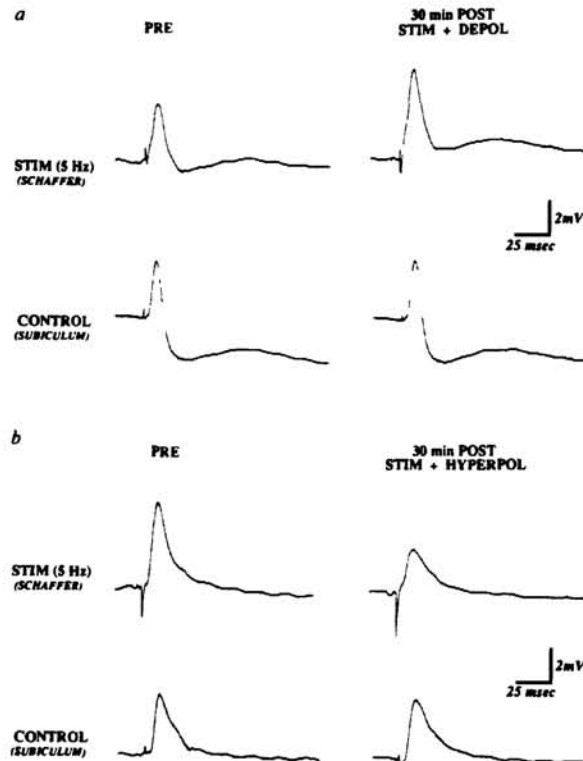

**Figure 4.** Pairing of postsynaptic hyperpolarization with stimulation of synapses on CA1 hippocampal pyramidal neurons produces LTD specific to the activated pathway, while pairing of postsynaptic depolarization with synaptic stimulation produces synapse-specific LTP. **a:** Intracellular evoked e.p.s.p.'s are shown at stimulated (**STIM**) and unstimulated (**CONTROL**) pathway synapses before (**Pre**) and 30 min after (**Post**) pairing a 20 mV depolarization (constant current +2.0 nA) with 5 Hz synaptic stimulation. The stimulated pathway exhibited associative **LTP** of the e.p.s.p., while the control, unstimulated input showed no change in synaptic strength. (RMP = -65 mV; $R_N$ = 35 M$\Omega$) **b:** Intracellular e.p.s.p.'s are shown evoked at stimulated and control pathway synapses before (**Pre**) and 30 min after (**Post**) pairing a 20 mV hyperpolarization (constant current -1.0 nA) with 5 Hz synaptic stimulation. The input (**STIM**) activated during the hyperpolarization showed associative **LTD** of synaptic evoked e.p.s.p.'s, while synaptic strength of the silent input (**CONTROL**) was unaltered. (RMP = -62 mV; $R_N$ = 38 M$\Omega$)

+64.0 -9.7%, n=4), while a control input inactive during the stimulation did not change (CONTROL), as reported previously (Kelso et al, 1986; Malinow and Miller, 1986; Gustaffson et al, 1987). Conversely, prolonged hyperpolarizing current injection paired with the same low-frequency stimuli led to induction of LTD in the stimulated pathway (Fig. 4b; STIM; -40.3 ± 6.3%, n=6), but not in the unstimulated pathway (CONTROL). The application of either depolarizing current, hyperpolarizing current, or the weak 5 Hz synaptic stimulation alone did not induce long-term alterations in synaptic strengths. Thus, hyperpolarization and simultaneous presynaptic activity supply sufficient conditions for the induction of LTD in CA1 pyramidal neurons.

## CONCLUSIONS

These experiments identify a novel form of anti-Hebbian synaptic plasticity in the hippocampus and confirm predictions made from modeling studies of information storage in neural networks. Unlike previous reports of synaptic depression in the hippocampus, the plasticity is associative, long-lasting, and is produced when presynaptic activity occurs while the postsynaptic membrane is hyperpolarized. In combination with Hebbian mechanisms also present at hippocampal synapses, associative LTP and associative LTD may allow neurons in the hippocampus to compute and store covariance between inputs (Sejnowski, 1977a,b; Stanton and Sejnowski, 1989). These finding make temporal as well as spatial context an important feature of memory mechanisms in the hippocampus.

Elsewhere in the brain, the receptive field properties of cells in cat visual cortex can be altered by visual experience paired with iontophoretic excitation or depression of cellular activity (Fregnac et al, 1988; Greuel et al, 1988). In particular, the chronic hyperpolarization of neurons in visual cortex coupled with presynaptic transmitter release leads to a long-term depression of the active, but not inactive, inputs from the lateral geniculate nucleus (Reiter and Stryker, 1988). Thus, both Hebbian and anti-Hebbian mechanisms found in the hippocampus seem to also be present in other brain areas, and covariance of firing patterns between converging inputs a likely key to understanding higher cognitive function.

This research was supported by grants from the National Science Foundation and the Office of Naval research to TJS. We thank Drs. Charles Stevens and Richard Morris for discussions about related experiments.

### References

Bienenstock, E., Cooper, L.N. and Munro, P. Theory for the development of neuron selectivity: orientation specificity and binocular interaction in visual cortex. *J. Neurosci.* **2**, 32-48 (1982).

Barrionuevo, G. and Brown, T.H. Associative long-term potentiation in hippocampal slices. *Proc. Nat. Acad. Sci. (USA)* **80**, 7347-7351 (1983).

Bliss, T.V.P. and Lomo, T. Long-lasting potentiation of synaptic transmission in the dentate area of the anaesthetized rabbit following stimulation of the perforant path. *J. Physiol. (Lond.)* **232**, 331-356 (1973).

Collingridge, G.L., Kehl, S.J. and McLennan, H. Excitatory amino acids in synaptic transmission in the Schaffer collateral-commissural pathway of the rat hippocampus. *J. Physiol. (Lond.)* **334**, 33-46 (1983).

Fregnac, Y., Shulz, D., Thorpe, S. and Bienenstock, E. A cellular analogue of visual cortical plasticity. *Nature (Lond.)* **333**, 367-370 (1988).

Greuel, J.M., Luhmann, H.J. and Singer, W. Pharmacological induction of use-dependent receptive field modifications in visual cortex. *Science* **242**, 74-77 (1988).

Gustafsson, B., Wigstrom, H., Abraham, W.C. and Huang, Y.Y. Long-term potentiation in the hippocampus using depolarizing current pulses as the conditioning stimulus to single volley synaptic potentials. *J. Neurosci.* **7**, 774-780 (1987).

Harris, E.W., Ganong, A.H. and Cotman, C.W. Long-term potentiation in the hippocampus involves activation of N-methyl-D-aspartate receptors. *Brain Res.* **323**, 132-137 (1984).

Kelso, S.R., Ganong, A.H. and Brown, T.H. Hebbian synapses in hippocampus. *Proc. Natl. Acad. Sci. USA* **83**, 5326-5330 (1986).

Kohonen, T. *Self-Organization and Associative Memory.* (Springer-Verlag, Heidelberg, 1984).

Larson, J. and Lynch, G. Synaptic potentiation in hippocampus by patterned stimulation involves two events. *Science* **232**, 985-988 (1986).

Levy, W.B. and Steward, O. Synapses as associative memory elements in the hippocampal formation. *Brain Res.* **175**, 233-245 (1979).

Levy, W.B. and Steward, O. Temporal contiguity requirements for long-term associative potentiation/depression in the hippocampus. *Neuroscience* **8**, 791-797 (1983).

Lynch, G.S., Dunwiddie, T. and Gribkoff, V. Heterosynaptic depression: a postsynaptic correlate of long-term potentiation. *Nature (Lond.)* **266**, 737-739 (1977).

Malinow, R. and Miller, J.P. Postsynaptic hyperpolarization during conditioning reversibly blocks induction of long-term potentiation *Nature (Lond.)* **320**, 529-530 (1986).

Mody, I., Stanton, P.K. and Heinemann, U. Activation of N-methyl-D-aspartate (NMDA) receptors parallels changes in cellular and synaptic properties of dentate gyrus granule cells after kindling. *J. Neurophysiol.* **59**, 1033-1054 (1988).

Reiter, H.O. and Stryker, M.P. Neural plasticity without postsynaptic action potentials: Less-active inputs become dominant when kitten visual cortical cells are pharmacologically inhibited. *Proc. Natl. Acad. Sci. USA* **85**, 3623-3627 (1988).

Sejnowski, T.J. and Tesauro, G. Building network learning algorithms from Hebbian synapses, in: *Brain Organization and Memory* J.L. McGaugh, N.M. Weinberger, and G. Lynch, Eds. (Oxford Univ. Press, New York, in press).

Sejnowski, T.J. Storing covariance with nonlinearly interacting neurons. *J. Math. Biology* **4**, 303-321 (1977).

Sejnowski, T. J. Statistical constraints on synaptic plasticity. *J. Theor. Biology* **69**, 385-389 (1977).

Stanton, P.K. and Sejnowski, T.J. Associative long-term depression in the hippocampus: Evidence for anti-Hebbian synaptic plasticity. *Nature (Lond.)*, in review.

Wigstrom, H. and Gustafsson, B. A possible correlate of the postsynaptic condition for long-lasting potentiation in the guinea pig hippocampus *in vitro. Neurosci. Lett.* **44**, 327-332 (1984).